# Adaptive Elastic Input Field for Recognition Improvement

**Minoru Asogawa**
C&C Research Laboratories, NEC
Miyamae, Miyazaki, Kawasaki Kanagawa 213 Japan
asogawa@csl.cl.nec.co.jp

## Abstract

For machines to perform classification tasks, such as speech and character recognition, appropriately handling deformed patterns is a key to achieving high performance. The authors presents a new type of classification system, an Adaptive Input Field Neural Network (AIFNN), which includes a simple pre-trained neural network and an elastic input field attached to an input layer. By using an iterative method, AIFNN can determine an optimal affine translation for an elastic input field to compensate for the original deformations. The convergence of the AIFNN algorithm is shown. AIFNN is applied for handwritten numerals recognition. Consequently, 10.83% of originally misclassified patterns are correctly categorized and total performance is improved, without modifying the neural network.

## 1 Introduction

For machines to accomplish classification tasks, such as speech and character recognition, appropriately handling deformed patterns is a key to achieving high performance [Simard 92] [Simard 93] [Hinton 92] [Barnard 91]. The number of reasonable deformations of patterns is enormous, since they can be either linear translations (an affine translation or a time shifting) or non-linear deformations (a set of combinations of partial translations), or both.

Although a simple neural network (e.g. a 3-layered neural network) is able to adapt

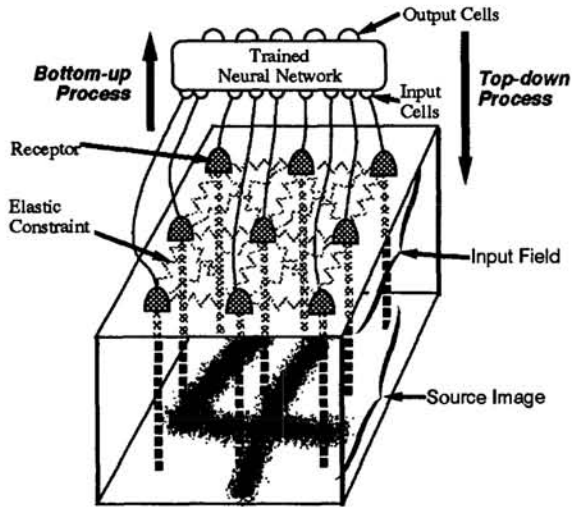

Figure 1: AIFNN

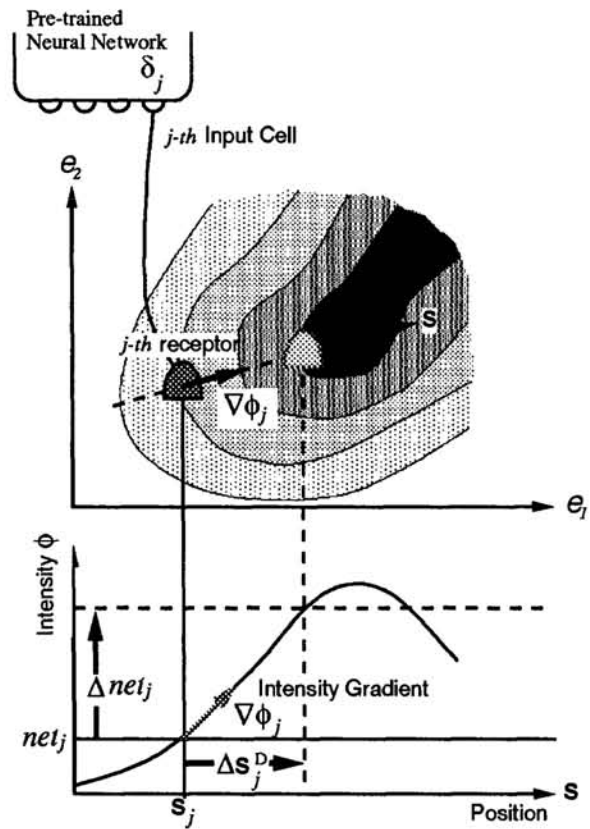

Figure 2: Delta Force

non-linear deformations and to discriminate noises, it is still necessary to have additional methods or data to appropriately process deformations.

This paper presents a new type of classification system, an Adaptive Input Field Neural Network (AIFNN), which includes a simple pre-trained neural network and an elastic input field attached to an input layer. The neural network is applied to non-linear deformation compensations and the elastic input field to linear deformations.

The AIFNN algorithm can determine an optimal affine translation for compensating for the original patterns' deformations, which are misclassified by the pre-trained neural network. As the result, those misclassified patterns are correctly classified and the final classification performance is improved, compared to that for the original neural network, without modifying the neural network.

## 2   Adaptive Input Field Neural Network (AIFNN)

AIFNN includes a pre-trained neural network and an elastic input field attached to an input layer (Fig. 1). The elastic input field contains receptors sampling input patterns at each location. Each receptor connects to a cell in the input layer. Each receptor links to its adjacent receptors with an elastic constraint and can move over

the input pattern independently, as long as its relative elastic constraint is satisfied. The affine translation of the whole receptor (e.g. a shift, rotation, scale and slant translation) satisfies an elastic constraint, since a constraint violation is induced by the receptors' relative locations. [1] Partial deformations are also allowed with a little constraint violation.

This feature of the elastic constraint is similar to that of the Elastic Net method [Durbin 87], which can solve NP-hard problems. Although this elastic net method is directly applicable to the template matching method, the performance is highly dependent on the template selection. Therefore, an elaborated feature space for non-linear deformations is mandatory [Hinton 92]. AIFNN utilizes something like an elastic net constraint, but does not require any prominent templates.

The AIFNN algorithm is a repeated sequence of a bottom-up process (calculating a guess and comparing with the presumption) and a top-down process (modifying receptor's location to decrease the error and to satisfy the input field constraints). For applying AIFNN as a classifier, a parallel search is performed; all categories are chosen as presumption categories and the AIFNN algorithm is executed. After hundreds of repetitions, an $L$ score is calculated, which is the sum of the error and the constraint violation in the elastic input field. A category which produces the lowest $L$ score is chosen as a plausible category. In Section 3, it is proved that all receptors will settle to an equilibrium state. In the following sections, details about the bottom-up and top-down processes are described.

**Bottom-Up Process:**
When a novel pattern is presented, each receptor samples activation corresponding to a pattern intensity at each location. Each receptor activation is directly transmitted to a corresponding neural network input cell. Those input values are forwarded through a pre-trained neural network and an output guess is obtained.

This guess is compared to the presumption category, and the negative of this error is defined as the presumption certainty. [2] For example, using the mean squared error criterion, the error $E^D$ is defined as follows;

$$E^D = \frac{1}{2}\sum_k (d_k - o_k)^2,$$ (1)

where $o_k$ is the output value, and $d_k$ is the desired value determined by the presumption category. The presumption certainty is defined as $-E^D$.

**Top-Down Process:**
To minimize the error and to maximize the presumption certainty, each receptor modifies the activation by moving its location over the input pattern. The new location for each receptor is determined by two elements; a direction which yields less error and a direction which satisfies the input field elastic constraint. The former element is called a Delta Force, since it relates to a delta value of an input layer cell. The latter element is named an Address Force. Each receptor moves to

a new location, which is determined by a sum of those two forces. The sum force is called the Combined Force. In the next two sections, details about these forces are described.

*Delta Force*: The Delta Force, which reduces $E^D$ by altering receptors' locations, is determined by two elements: a partial derivative for the input value to the error, and a local pattern gradient at each receptor location (Fig. 1).

To decrease the error $E^D$, the value divergence for the $j$-th cell is computed as,

$$\Delta net_j \equiv -\alpha^D \frac{\partial E^D}{\partial net_j} = \alpha^D \delta_j, \tag{2}$$

where $\alpha^D$ is small positive number and $\delta_j$ is a delta value for the $j$-th input cell and computed by the back-propagation [Yamada 91]. $\Delta net_j$ and a local pattern gradient $\nabla \phi_j$ are utilized to calculate a Delta Force $\Delta s_j^D$; a scalar value of $\Delta s_j^D$ is given as,

$$|\Delta s_j^D| = \frac{\Delta net_j}{|\nabla \phi_j|}. \tag{3}$$

The direction of the Delta Force $\Delta s_j^D$ is chosen as being parallel to that of $\nabla \phi_j$. Consequently, $\Delta s_j^D$ is given as,

$$\Delta s_j^D = \frac{\Delta net_j}{|\nabla \phi_j|} \frac{\nabla \phi_j}{|\nabla \phi_j|} = \alpha^D \frac{\delta_j}{|\nabla \phi_j|^2} \nabla \phi_j. \tag{4}$$

To avoid $\Delta s_j^D$ becoming infinity, when $|\nabla \phi_j|$ is almost equal to 0, a small constant $c(= \frac{1}{4})$ is added to the denominator; therefore, $\Delta s_j^D$ is defined as,

$$\Delta s_j^D = \alpha^D \frac{\delta_j}{|\nabla \phi_j|^2 + c} \nabla \phi_j. \tag{5}$$

*Address Force*: If each receptor is moved iteratively following only the Delta Force, the error becomes its minimum. However, receptors may not satisfy the input field constraint and induce a large constraint violation $E^A$. Here, $E^A$ is defined by a distance between a receptor's lattice $\mathbf{S}$ and a lattice which is derived by an affine translation from the original lattice. Therefore, $E^A$ is defined as follows;

$$E^A \equiv \frac{1}{2} d(\mathbf{S^N}, \mathbf{S}) = \frac{1}{2} \sum_i \|s_i^N - s_i\|^2$$

$$= \frac{1}{2} d(\mathcal{T}(\mathbf{S^O}; \mathbf{t}), \mathbf{S}), \tag{6}$$

where $d(\cdot, \cdot)$ is a distance measure for two receptor's lattices. $\mathbf{S}$ is a current receptor lattice. $\mathbf{S^N}$ is the receptor lattice given by the affine translation $\mathcal{T}(\cdot)$ with parameters $\mathbf{t}$ and $\mathbf{S^O}$. $\mathbf{S^O}$ is the original receptor lattice.

Therefore, as long as the receptor's lattice can be driven by some affine translation, there is no constraint violation.

The affine parameters $\mathbf{t}$ are estimated so as to minimize $E^A$;

$$\frac{\partial E^A}{\partial t_i} = 0 \qquad \text{for } i = 1, \cdots, 6. \tag{7}$$

Since $E^A$ is quadratic with respect to $t_i$, computing $t_i$ is moderate. The Address Force for the $j$-th receptor $\Delta s_j^A$ is defined as the partial derivative to $E^A$ with respect to the receptor's location $s_j$;

$$\Delta s_j^A \equiv -\alpha^A \frac{\partial E^A}{\partial s_j}, \tag{8}$$

where $\alpha^A$ is a small positive constant.

*Combined Force*: Here, all receptors are moved by a Combined Force $\Delta s$, which is a sum of the Delta Force $\Delta s^D$ and the Address Force $\Delta s^A$.

After one hundred iterations, all receptors are moved to the location which produces the minimum output error and the minimum constraint violation. Final states are evaluated with a new measurement $L$ score, which is the sum of the error $E^D$ and the constraint violation $E^A$; i.e. $L = E^D + E^A$.

This $L$ score is utilized to choose the correct category in a parallel search. In a parallel search, each category is temporarily chosen as a presumption and converged $L$ scores are calculated. Those scores are compared and the category yielding the smallest $L$ score is chosen as the correct category. This method fully exploits the features of AIFNN, but it requires a large amount of computation, which can fortunately be processed totally in parallel. In the following section, convergence of the AIFNN is shown.

## 3   Convergence

Convergence is shown by proving that the $L$ is a Lyapunov function. When the $L$ is a Lyapunov function, all receptors converge to some locations after iterations. The necessary and sufficient conditions for a Lyapunov function are (1) $L$ has a lower bound and (2) $L$ monotonically decreases by applying the Combined Forces.

**(1) Lower Bound:**
$E^D$ is the squared error at the output layer. Therefore, $E^D \geq 0$. $E^A$ is the constraint violation, which is defined with a distance between two lattices. Therefore, $E^A \geq 0$. Since the $L$ is a sum of $E^D$ and $E^A$, the existence of a lower bound for the $L$ is proved. □

**(2) Monotonically Decrease:**
The derivative of the $L$ is calculated to show that the $L$ decreases monotonically.

$$\begin{aligned}
\frac{d L}{d t} &= \frac{d E^D}{d t} + \frac{d E^A}{d t} \\
&= \sum_i \left\{ \frac{\partial E^D}{\partial s_i} \frac{d s_i}{d t} \right\} + \sum_i \left\{ \frac{\partial E^A}{\partial s_i} \frac{d s_i}{d t} \right\} \\
&= \sum_i \left\{ \left( \frac{\partial E^D}{\partial s_i} + \frac{\partial E^A}{\partial s_i} \right) \frac{d s_i}{d t} \right\},
\end{aligned} \tag{9}$$

where $\dfrac{d s_i}{d t}$ is the Combined Force and given as,

$$\frac{d s_i}{d t} = \Delta s^D + \Delta s^A. \tag{10}$$

When a source image is smooth and $|\nabla \phi_i|$ is smaller than $c$, the following approximation is satisfied;

$$\frac{\nabla \phi_i}{|\nabla \phi_i|^2 + c} \simeq \nabla \phi_i. \qquad (11)$$

By using Eq. (11), the Delta Force is approximated as follows;

$$\Delta \mathbf{s}^D = \alpha^D \frac{\nabla \phi_i}{|\nabla \phi_i|^2 + c} \frac{\delta_i}{\nabla \phi_i} \simeq -\alpha^D \frac{\partial E^D}{\partial s_i}. \qquad (12)$$

By using Eqs. (8) and (12), and by letting $\alpha^D = \alpha^A$, the $L$ derivative is computed as follows;

$$\frac{d L}{d t} \simeq -\alpha^A \sum_i \left( \frac{\partial E^D}{\partial \mathbf{s}_i} + \frac{\partial E^A}{\partial \mathbf{s}_i} \right)^2 \leq 0. \qquad (13)$$

With Eq. (13), it is proved that $L$ decreases monotonically.□

## 4   Experiments and Results

Hand-written numerals recognition is chosen as one of the applications of AIFNN, since performance improvement is shown by compensating for deformations [Simard 92] [Simard 93] [Hinton 92]. The numeral inputs are bi-level images of $32 \times 40$. They are blurred with a $5 \times 5$ Gaussian kernel and resampled to $14 \times 18$ pixel gray level images. To calculate an intensity and a local gradient between grids, bi-linear Lagrange interpolation is utilized.

A neural network is 3 layered. The numbers of cells for the input layer, the hidden layer and the output layer are 252, 20 and 10, respectively. To obtain a simpler weight configuration, two techniques are utilized; a constant weight decay [Ishikawa 89] and a small constant addition to output function derivatives [Fahlman 88]. Training is repeated for 180 epochs with 2500 numerals, and tested with another 2500. Since image edges are almost blank, about 2400 connections between the input layer and the hidden layer are equal to 0; therefore, the number of parameters is reduced to 2870.

In this experiment, a simple decision method is used; the maximum output cell is chosen as a guess and patterns are rejected when the error of the guess is greater than a threshold value. Naturally, a low threshold yields a low misclassification rate, but also yields a high rejection rate [Martin 92]. With the maximum threshold, the rates of rejection, correct classification and misclassification are 0.00%(0 patterns), 95.20%(2380 patterns) and 4.80%(120 patterns), respectively. For the 2500 numerals learning data, these rates are 0.00%(0 patterns), 99.40%(2485 patterns) and 0.60%(15 patterns). When a threshold is 0.001, the rates of rejection, correct classification and misclassification are 43.52%(1088 patterns), 56.40%(1410 patterns) and 0.08%(2 patterns), respectively.

AIFNN is applied to these 1088 rejected patterns. and classifies 997 patterns correctly. Therefore, total performances for rejection, correct classification and misclassification become 0.00%(0 patterns), 95.72%(2393 patterns) and 4.28%(107 patterns), respectively. As the classification performance is improved; the number of

misclassified patterns reduces from 120 to 107 without modifying the neural network. 10.83% of the originally misclassified patterns are correctly categorized. Fig. 3 shows an input field after one hundred iterations.

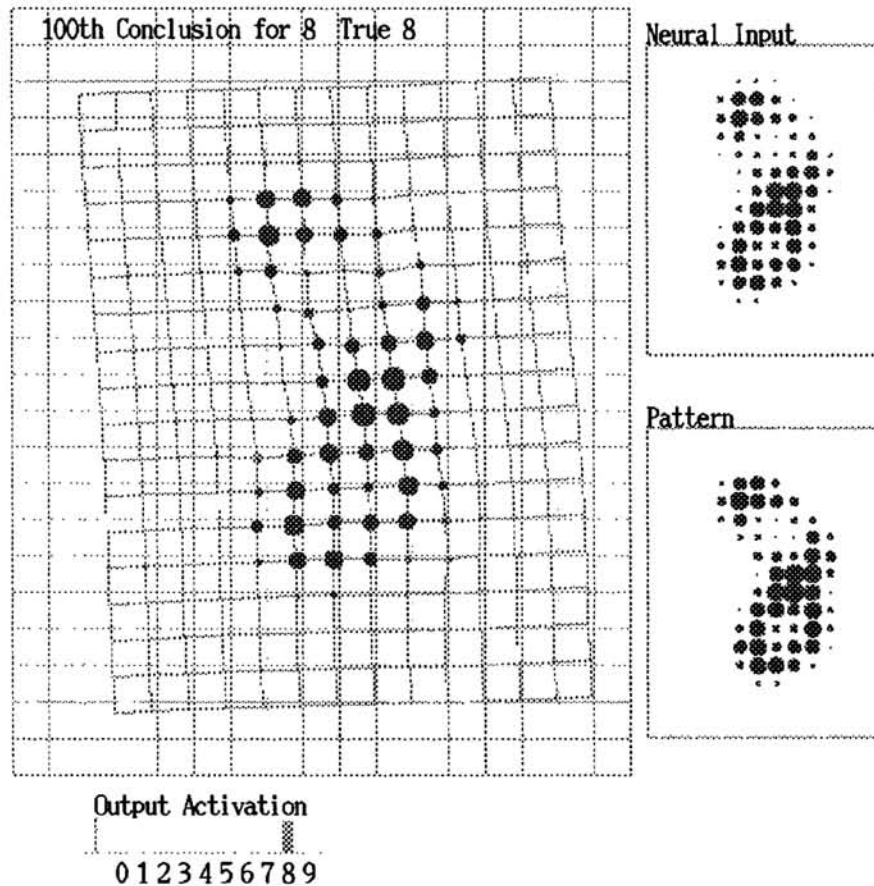

Output Activation

0 1 2 3 4 5 6 7 8 9

In the figure on the left, receptors are located at each grid point in a gray lattice. The circle diameter corresponds to the pattern intensity at each receptor's location. The bottom right figure indicates the source image, and the top right figure indicates the neural network input. This image was initially misclassified as 3 instead of 8. After iteration with presumption as 8, category 8 gets the highest activation and the receptor's lattice is rotated to compensate for the initial deformation.

Figure 3: Input Field After Adaptation

## 5   Discussion

It is shown that the AIFNN can improve the classification performance for the original neural network, without modifications. This performance improvement is caused by an optimal affine translations estimation for rejected patterns.

Although an affine translation is discussed in this paper, the algorithm is applicable to any deformation mechanism; such as a gain and offset equalization and 3D perspective deformation.

The requirement for a neural network in AIFNN is the capability of calculating partial derivatives for an input layer, so a layered neural network is utilized in this paper. Since partial derivative can be computed by numerical approximation, practically any neural network is applicable for AIFNN. Moreover, any differentiable error criterion is applicable; such as, a KL information and a likelihood.

To reduce computation, a sequential searching is also possible; a presumption is chosen as the most plausible category, e.g. the smallest error category. If the $L$ score falls behind a threshold, this presumption is regarded as correct. If it's not, another plausible category is chosen as a presumption and tested [Asogawa 91].

## Footnotes

[1] In previous papers, [Asogawa 90] and [Asogawa 91], a shift and rotation translation was taken into account. In those models, a scale and slant translation violated the elastic constraint.

[2] Although another category coding schema is also possible, for simplicity, it is presumed that each output cell corresponds to one certain category.

# References

[Asogawa 90] M. Asogawa, "Adaptive Input Field Neural Network – that can recognize rotated and/or shifted character –", *Proceedings of IJCNN '90 at San Diego*, vol. 3. pp. 733-738. June 1990.

[Asogawa 91] M. Asogawa, "Adaptive Input Field Neural Network", *Proceedings of IJCNN '91 at Singapore*, vol. 1. pp. 83-88. November 1991.

[Barnard 91] E. Barnard et al., "Invariance and Neural Nets", *IEEE trans. on Neural Networks*, vol. 2. no. 5, pp. 498-508. 1992.

[Durbin 87] R. Durbin et al., "An analogue approach to the traveling salesman problem using an elastic net method", *Nature*, vol. 326. pp. 689-691. 1987.

[Fahlman 88] S. Fahlman, "An empirical study of learning speed in back-propagation networks", CMU-CS-88-162, 1988.

[Hinton 92] G.E. Hinton et al., "Adaptive Elastic Models for Hand-Printed Character Recognition", *Advances in Neural Information Processing Systems*, vol. 4. pp. 512-519. 1992.

[Ishikawa 89] M. Ishikawa, "A structural learning algorithm with forgetting of link weights", *Proceedings of IJCNN '89 at Washington DC.*, vol. 2, pp. 626, 1989.

[Martin 92] G. L. Martin et al., "Recognizing Overlapping Hand-Printed Characters by Centered-Object Integrated Segmentation and Recognition", *Advances in Neural Information Processing Systems*, vol. 4. pp. 504-511. 1992.

[Simard 92] P. Simard et al., "Tangent Prop - A Formalism for Specifying Selected Invariances in an Adaptive Network", *Advances in Neural Information Processing Systems*, vol. 4. pp. 895-903. 1992.

[Simard 93] P. Simard et al., "Efficient Pattern Recognition Using a New Transformation Distance", *Advances in Neural Information Processing Systems*, vol. 5. pp. 50-58. 1993.

[Yamada 91] K. Yamada, "Learning of category boundaries based on inverse recall by multilayer neural network", *Proceedings of IJCNN '91 at Seattle*, pp. 7-12 vol.2 1991.